# Semigroup Kernels on Finite Sets

**Marco Cuturi**
Computational Biology Group
Ecole des Mines de Paris
35 rue Saint Honoré
77300 Fontainebleau
marco.cuturi@ensmp.fr

**Jean-Philippe Vert**
Computational Biology Group
Ecole des Mines de Paris
35 rue Saint Honoré
77300 Fontainebleau
jean-philippe.vert@ensmp.fr

## Abstract

Complex objects can often be conveniently represented by finite sets of simpler components, such as images by sets of patches or texts by bags of words. We study the class of positive definite (p.d.) kernels for two such objects that can be expressed as a function of the merger of their respective sets of components. We prove a general integral representation of such kernels and present two particular examples. One of them leads to a kernel for sets of points living in a space endowed itself with a positive definite kernel. We provide experimental results on a benchmark experiment of handwritten digits image classification which illustrate the validity of the approach.

## 1 Introduction

Suppose we are to deal with complex (e.g non-vectorial) objects from a set $\mathcal{Z}$ on which we wish to apply existing kernel methods [1] to perform tasks such as classification or regression. Assume furthermore that the latter objects can be meaningfully described by small components contained in a set $\mathcal{X}$. Namely, we suppose that we can define an a priori mapping $\tau$ which maps any $z \in \mathcal{Z}$ into a finite unordered list of elements of $\mathcal{X}$, $\tau(z) = [x_1, x_2, ..., x_n]$, through a sampling process which may be exhaustive, heuristic or random both in the quantity of sampled components $n$ and in the way those components are extracted. Comparing two such complex objects through the direct comparison of their respective lists of components has attracted much attention recently, namely through the definition of p.d. kernel on such $\tau$-lists. Most recent approaches to compare two $\tau$-lists involve the estimation of two distributions $p_z$ and $p_{z'}$ on $\mathcal{X}$ within a parametric class of models that fit (e.g. in maximum likelihood (ML) sense) respectively $\tau(z)$ and $\tau(z')$ seen as a samples from laws on $\mathcal{X}$, where each resulting law could be identified with $z$ and $z'$ respectively. Such a kernel is then defined between $p_z$ and $p_{z'}$, as seen for example in [2] with the Information Diffusion kernel, in [3] with the family of Mutual Information Kernels or in [4] with the use of the Battacharyya affinity between $p_z$ and $p_{z'}$. An alternative and non-parametric approach to $\tau$-lists comparison that studies the subspaces generated by points of $\tau(z)$ and $\tau(z')$ in a feature space was also proposed in [5], recalling elements presented in Kernel-Canonical Correlation Analysis [6].

We explore in this contribution a different direction to kernel design for lists by studying the class of kernels whose value computed on two lists is only defined through its value on their

concatenation. This approach was already used in [7], where a particular kernel for strings that only compares two strings through their concatenation is presented. In this paper, the approach is extended to a more general and abstract setting of $\tau$-lists, but the motivation remains the same as in [7]: if two $\tau$-lists are similar, e.g. in terms of the distribution of the components they describe, then their concatenation will be more "concentrated" than if they are very different, in which case it might look more like a reunion of two disjoint sets of points. As a result, one can expect to get a relevant measure of similarity, and hence a kernel, by studying properties of the concatenation of two lists such as its concentration.

After an example of a valid kernel for lists seen as measures on the space of components (Section 2), we provide a complete characterization for this class of kernels (Section 3) by casting them in the context of semigroup kernels. This leads to the definition of a second kernel based on exponential densities on $\mathcal{X}$, which boils down after a numerical approximation to the computation of the entropy of the maximum likelihood density (taken in the considered exponential family) of the points numbered by lists of components. This kernel is extended in Section 4 to points taken in a reproducing kernel Hilbert space defined by a kernel $\kappa$ on $\mathcal{X}$, and is then tested on a problem of image classification, where images are seen as bags of pixels and a non-linear kernel between pixels is used (Section 5).

## 2   The entropy kernel

As a warm-up, let us assume that the set $\mathcal{X}$ is measurable, e.g. $\mathcal{X} = \mathbb{R}^d$, and that to any point $x \in \mathcal{X}$ we can associate a probability measure on $\mathcal{X}$ with density $\mu_x$ with respect to a common measure (e.g. the Borel uniform measure), with finite entropy $h(\mu) \stackrel{\text{def}}{=} -\int_{\mathcal{X}} \mu \ln \mu$. Consider for example a Gaussian distribution with mean $x$ and fixed variance. A natural way to represent an unordered list $\tau(z) = [x_1, x_2, ..., x_n] \in \mathcal{X}^n$ is by the density $\mu_\tau = 1/n \sum_{i=1}^{n} \mu_{x_i}$. In that case, a p.d. kernel $k$ between unordered lists $\tau$ and $\tau'$ that only depends on their concatenation $\tau(z) \cdot \tau(z')$ is equivalent to a p.d. kernel between densities $\mu$ and $\mu'$ that only depends on $\mu + \mu'$. Hence we are looking for a p.d. kernel on the set $\mathcal{P}$ of probability densities of finite entropy of the form $\kappa(\mu, \mu') = \varphi(\mu + \mu')$. An example of such a kernel is provided in the following proposition. Recall that a negative definite (n.d.) kernel on a set $X$ is a symmetric function $g : X^2 \to \mathbb{R}$ that satisfies $\sum_{i,j=1}^{n} c_i c_j g(x_i, x_j) \le 0$ for any $n \in \mathbb{N}, (x_1, \ldots, x_n) \in X^n$, and $(c_1 \ldots, c_n) \in \mathbb{R}^n$ with $\sum_{i=1}^{n} c_i = 0$. A useful link between p.d. and n.d. kernels is that $g$ is n.d. if and only if $\exp(-tg)$ is p.d. for all $t > 0$ [8, Theorem 3.2.2.].

**Proposition 1.** *The function* $g : \mu, \mu' \mapsto h(\frac{\mu+\mu'}{2})$ *is negative definite on* $\mathcal{P}$, *making* $k_h(\mu, \mu') \stackrel{\text{def}}{=} e^{-th(\frac{\mu+\mu'}{2})}$ *a p.d. kernel on* $\mathcal{P}$ *for any* $t > 0$. *We call* $k_h$ *the entropy kernel between two measures.*

The entropy kernel is already a satisfactory answer to our initial motivation to look at merger of points. Observe that if $\mu_x$ is a probability density around $x$, then $\mu_\tau$ can often be thought of as an estimate of the distribution of the points in $\tau$, and $(\mu_\tau + \mu_{\tau'})/2$ is an estimate of the distribution of the points enumerated in $\tau \cdot \tau'$. If the latter estimate has a small entropy we can guess that the points in $\tau$ and $\tau'$ are likely to have similar distributions which is exactly the similarity that is quantified by the entropy kernel.

*Proof of Proposition 1.* It is known that the real-valued function $r : y \mapsto -y \ln y$ is n.d. on $\mathbb{R}_+$ as a semigroup endowed with addition [8, Example 6.5.16]. As a consequence the function $f \mapsto r \circ f$ is n.d. on $\mathcal{P}$ as a pointwise application of $r$, and so is its summation on $\mathcal{X}$. For any real-valued n.d. kernel $k$ and any real-valued function $g$, we have trivially that $(y, y') \mapsto k(y, y') + g(y) + g(y')$ remains negative definite, hence $h(\frac{f+f'}{2})$ is n.d. through $h(\frac{f+f'}{2}) = \frac{1}{2}h(f + f') + \frac{\ln 2}{2}(|f| + |f'|)$, yielding positive definiteness of $k_h$. $\square$

# 3 Semigroups and integral representations of p.d. kernels on finite Radon measures

In order to generalize the example presented in the previous section, let us briefly recall the concept of p.d. kernels on semigroups [8]. A nonempty set $\mathcal{S}$ is called an Abelian (autoinvolutive) semigroup if it is equipped with an *Abelian associative composition* $\circ$ admitting a neutral element in $\mathcal{S}$. A function $\varphi : \mathcal{S} \mapsto \mathbb{R}$ is called a *positive definite* (resp. *negative definite*) *function* on the semigroup $(S, \circ)$ if $(s, t) \mapsto \varphi(s \circ t)$ is a p.d. (resp. n. d.) kernel on $\mathcal{S} \times \mathcal{S}$.

The entropy kernel defined in Proposition 1 is therefore a p.d. kernel on the semigroup of measures with finite entropy endowed with usual addition. This can be generalized by assuming that $\mathcal{X}$ is a Hausdorff space, which suffices to consider the set of finite Radon measures $M_+^b(\mathcal{X})$ [8]. For $\mu \in M_+^b(\mathcal{X})$, we note $|\mu| = \mu(\mathcal{X}) < +\infty$. For a Borel measurable function $f \in \mathbb{R}^{\mathcal{X}}$, we note $\mu[f] = \int_{\mathcal{X}} f d\mu$. Endowed with the usual Abelian addition between measures, $(M_+^b(\mathcal{X}), +)$ is an Abelian semigroup. The reason to consider this semigroup is that there is a natural semigroup homomorphism between finite lists of points and elements of $M_+^b(\mathcal{X})$ given by $\tau = [x_1, ..., x_n] \mapsto \mu_\tau = \sum_{i=1}^{n} \mu_{x_i}$, where $\mu_x \in M_+^b(\mathcal{X})$ is an arbitrary finite measure associated with each $x \in \mathcal{X}$. We discussed in section 2 the case where $\mu_x$ has a density, but more general measures are allowed, such as $\mu_x = \delta_x$, the Dirac measure. Observe that when we talk about lists, it should be understood that some objects might appear with some multiplicity which should be taken into account (specially when $\mathcal{X}$ is finite), making us consider weighted measures $\mu = \sum_{i=1}^{n} c_i \mu_{x_i}$ in the general case. We now state the main result of this section which characterizes bounded p.d. functions on the semigroup $M_+^b(\mathcal{X})$,

**Theorem 1.** *A bounded real-valued function $\varphi$ on $M_+^b(\mathcal{X})$ such that $\varphi(0) = 1$ is p.d. if and only if it has an integral representation:*

$$\varphi(\mu) = \int_{C^+(\mathcal{X})} e^{-\mu[f]} d\nu(f),$$

*where $\nu$ is a uniquely determined positive radon measure on $C^+(\mathcal{X})$, the space of non-negative-valued continuous functions of $\mathbb{R}^{\mathcal{X}}$ endowed with the topology of pointwise convergence.*

*Proof.* (sketch) Endowed with the topology of weak convergence, $M_+^b(\mathcal{X})$ is a Hausdorff space [8, Proposition 2.3.2]. The general result of integral representation of bounded p.d. function [8, Theorem 4.2.8] therefore applies. It can be shown that bounded semicharacters on $M_+^b(\mathcal{X})$ are exactly the functions of the form $\mu \mapsto \exp(-\mu[f])$ where $f \in C^+(\mathcal{X})$ by using the characterization of semicharacters on $(\mathbb{R}_+, +)$ [8, Theorem 6.5.8] and the fact that atomic measures is a dense subset of $M_+^b(\mathcal{X})$ [8, Theorem 2.3.5]. $\square$

As a constructive application to this general representation theorem, let us consider the case $\mu_x = \delta_x$ and consider, as a subspace of $C^+(\mathcal{X})$, the linear span of $N$ non-constant, continuous, real-valued and linearly independent functions $f_1, ..., f_N$ on $\mathcal{X}$. As we will see below, this is equivalent to considering a set of densities defined by an exponential model, namely of the form $p_\theta(x) = \exp(\sum_{j=1}^{N} \theta^j f_j(x) - \psi(\theta))$ where $\theta = (\theta^j)_{j=1..N} \in \Theta \subset \mathbb{R}^N$ is variable and $\psi$ is a real-valued function defined on $\Theta$ to ensure normalization of the densities $p_\theta$. Considering a prior $\omega$ on the parameter space $\Theta$ is equivalent to defining a Radon measure taking positive values on the subset of $C^+(\mathcal{X})$ spanned by functions $f_1, ..., f_N$. We now have ( see [9] for a geometric point of view) that:

**Theorem 2.** $\hat{\theta}_\mu \in \Theta$ *being the ML parameter associated with $\mu$ and noting $p_\mu = p_{\hat{\theta}_\mu}$,*

$$\varphi_\omega(\mu) = e^{-|\mu|h(p_\mu)} \int_\Theta e^{-|\mu|d(p_\mu||p_\theta)} \omega(d\theta),$$

*is a p.d. kernel on the semigroup of measures, where $d(p||q) = \int_{\text{supp}(q)} p \ln \frac{p}{q}$ is the Kullback-Leibler divergence between $p$ and $q$.*

Although an exact calculation of the latter equation is feasible in certain cases (see [10, 7]), an approximation can be computed using Laplace's approximation. If for example the prior on the densities is taken to be Jeffrey's prior [9, p.44] then the following approximation holds:

$$\varphi(\mu) \underset{|\mu| \to \infty}{\sim} \tilde{\varphi}(\mu) := e^{-|\mu|h(p_\mu)} \left( \frac{2\pi}{|\mu|} \right)^{\frac{N}{2}}. \tag{1}$$

The ML estimator being unaffected by the total weight $|\mu|$, we have $\tilde{\varphi}(2\mu) = \tilde{\varphi}(\mu)^2 (\frac{|\mu|}{4\pi})^{\frac{N}{2}}$ which we use to renormalize our kernel on its diagonal:

$$k(\mu, \mu') = \frac{e^{-(|\mu+\mu'|)h(p_{\mu+\mu'})}}{e^{-|\mu|h(p_\mu)-|\mu'|h(p_{\mu'})}} \left( \frac{2\sqrt{|\mu||\mu'|}}{|\mu|+|\mu'|} \right)^{\frac{N}{2}}$$

Two problems call now for a proper renormalization: First, if $|\mu'| \ll |\mu|$ (which would be the case if $\tau$ describes far more elements than $\tau'$), the entropy $h(p_{\mu+\mu'})$ will not take into account the elements enumerated in $\mu'$. Second, the value taken by our p.d function $\tilde{\varphi}$ decreases exponentially with $|\mu|$ as can be seen in equation (1). This inconvenient scaling behavior leads in practice to bad SVM classification results due to diagonal dominance of the Gram matrices produced by such kernels (see [11] for instance). Recall however that the Laplace approximation can be accurate only when $|\mu| \gg 0$. To take into account this tradeoff on the ideal range of $|\mu|$, we rewrite the previous expression using a common width parameter $\beta$ after having applied a renormalization on $\mu$ and $\mu'$:

$$k_\beta(\mu, \mu') = k(\frac{\beta}{|\mu|}\mu, \frac{\beta}{|\mu'|}\mu') = e^{-2\beta \left( h(p_{\mu''}) - \frac{h(p_\mu)+h(p_{\mu'})}{2} \right)}, \tag{2}$$

where $\mu'' = \frac{\mu}{|\mu|} + \frac{\mu'}{|\mu'|}$. $\beta$ should hence be big enough in practical applications to ensure the consistency of Laplace's approximation and thus positive definiteness, while small enough to avoid diagonal dominance. We will now always suppose that our atomic measures are normalized, meaning that their total weight $\sum_{i=1}^n c_i$ always sums up to 1.

Let us now review a practical case when $\mathcal{X}$ is $\mathbb{R}^k$, and that some kind of gaussianity among points makes sense. We can use $k$-dimensional normal distributions $p_{m,\Sigma} \sim \mathcal{N}(m, \Sigma)$ (where $\Sigma$ is a $k \times k$ p.d. matrix) to define our densities. The ML parameters of a measure $\mu$ are in that case : $\bar{\mu} = \sum_{i=1}^n c_i x_i$ and $\Sigma_\mu = \sum_{i=1}^n c_i(x_i - \bar{\mu})(x_i - \bar{\mu})^\top$. Supposing that the span of the $n$ vectors $x_i$ covers $\mathbb{R}^k$ yields non-degenerated covariance matrices. This ensures the existence of the entropy of the ML estimates through the formula [12]: $h(p_{m,\Sigma}) = \frac{1}{2} \ln \left( (2\pi e)^n |\Sigma| \right)$. The value of the normalized kernel in (2) is then:

$$k_\beta(\mu, \mu') = \left( \frac{\sqrt{|\Sigma_\mu||\Sigma_{\mu'}|}}{|\Sigma_{\mu''}|} \right)^{2\beta}.$$

This framework is however limited to vectorial data for which the use of Gaussian laws makes any sense. An approach designed to bypass this double restriction is presented in the next section, taking advantage of a prior knowledge on the components space through the use of a kernel $\kappa$.

## 4 A kernel defined through regularized covariance operators

Endowing $\mathcal{X}$ (now also considered 2-separable) with a p.d. kernel $\kappa$ bounded on the diagonal, we make use in this section of its corresponding reproducing kernel Hilbert space (RKHS, see [13] for a complete survey). This RKHS is denoted by $\Xi$, and its feature map by $\xi : x \mapsto \kappa(x, \cdot)$. $\Xi$ is infinite dimensional in the general case, preventing any systematical use of exponential densities on that feature space. We bypass this issue through a generalization of the previous section by still assuming some "gaussianity" among the elements numbered by atomic measures $\mu, \mu'$ and $\mu''$ which, once mapped in the feature space, are now functions. More precisely, our aim when dealing with Euclidean spaces was to estimate finite dimensional covariance matrices $\Sigma_\mu, \Sigma_{\mu'}, \Sigma_{\mu''}$ and compare them in terms of their spectrum or more precisely through their determinant. In this section we use such finite samples to estimate, diagonalize and regularize three covariance operators $S_\mu, S_{\mu'}, S_{\mu''}$ associated with each measure on $\Xi$, and compare them by measuring their respective dispersion in a similar way. We note for $\xi \in \Xi$ its dual $\xi^*$ (namely the linear form $\Xi \to \mathbb{R}$ s.t. $\zeta \mapsto \xi^* \zeta = \langle \xi, \zeta \rangle_\Xi$) and $||\xi||^2 = \xi^* \xi$. Let $(e_i)_{i \in \mathbb{N}}$ be a complete orthonormal base of $\Xi$ (i.e. such that $\overline{\text{span}}(e_i)_{i \in \mathbb{N}} = \Xi$ and $e_i^* e_j = \delta_{ij}$). Given a family of positive real numbers $(t_i)_{i \in \mathbb{N}}$, we note $S_{t,e}$ the bilinear symmetric operator which maps $\xi, \zeta \mapsto \xi^* S_{t,e} \zeta$ where $S_{t,e} = \sum_{i \in \mathbb{N}} t_i e_i e_i^*$.

For an atomic measure $\mu$ and noting $\tilde{\xi}_i \overset{\text{def}}{=} (\xi_i - \mu[\xi])$ its $n$ centered points in $\Xi$, the empirical covariance operator $S_\mu = \sum_{i=1}^n c_i \tilde{\xi}_i \tilde{\xi}_i^*$ on $\Xi$ can be described through such a diagonal representation by finding its principal eigenfunctions, namely orthonormal functions in $\Xi$ which maximize the expected (w.r.t to $\mu$) variance of the normalized dot-product $h_v(\xi) \overset{\text{def}}{=} \frac{v^* \xi}{||v||}$ here defined for any $v$ of $\Xi$. Such functions can be obtained through the following recursive maximizations:

$$v_j = \underset{v \in \Xi, v \perp \{v_1, \ldots, v_{j-1}\}}{\text{argmax}} \text{var}_\mu(h_v(\xi)) = \underset{v \in \Xi, v \perp \{v_1, \ldots, v_{j-1}\}}{\text{argmax}} \frac{1}{||v_j||^2} \sum_{i=1}^n c_i v_j^* \tilde{\xi}_i.$$

As in the framework of Kernel PCA [1] (from which this calculus only differs by considering weighted points in the feature space) we have by the representer theorem [1] that all the solutions of these successive maximizations lie in $\text{span}(\{\tilde{\xi}_i\}_{i=1..n})$. Thus for each $v_j$ there exists a vector $\alpha_j$ of $\mathbb{R}^n$ such that $v_j = \sum_{i=1}^d \alpha_{j,i} \tilde{\xi}_i$ with $||v_j||^2 = \alpha_j^\top \tilde{K}_\mu \alpha_j$ where $\tilde{K}_\mu = (I_n - \mathbb{1}_{n,n} \Delta_c) K_\mu (I_n - \Delta_c \mathbb{1}_{n,n})$ is the centered Gram matrix $K_\mu = [\kappa(x_i, x_j)]_{1 \le i,j \le n}$ of the points taken in the support of $\mu$, with $\mathbb{1}_{n,n}$ being the $n \times n$ matrix composed of ones and $\Delta_c$ the $n \times n$ diagonal matrix of $c_i$ coefficients. Our latter formulation is however ill-defined, since any $\alpha_j$ is determined up to the addition of any element of $\ker \tilde{K}_\mu$. We thus restrict our parameters $\alpha$ to lie in $E \overset{\text{def}}{=} \ker \tilde{K}_\mu^\perp \subset \mathbb{R}^n$ to consider functions of positive squared norm, having now:

$$\alpha_j = \underset{\alpha \in E : \forall k < j, \alpha^\top \tilde{K} \alpha_k = 0}{\text{argmax}} \frac{\alpha^\top \tilde{K}_\mu \Delta_c \tilde{K}_\mu \alpha}{\alpha^\top \tilde{K}_\mu \alpha} \quad \left( = \text{var}_\mu(h_{v_j}(\xi)) \right)$$

Both endomorphism $\tilde{K}_\mu \Delta_c \tilde{K}_\mu$ and $\tilde{K}_\mu$ being symmetric *positive* definite on $E$ (one can easily prove that $\ker \tilde{K}_\mu = \ker \tilde{K}_\mu \Delta_c \tilde{K}_\mu$), the right-hand argument of the previous equation, known as the Rayleigh quotient of $\tilde{K}_\mu \Delta_c \tilde{K}_\mu$ by $\tilde{K}_\mu$, can be maximized through a Hermitian generalized eigenvalue decomposition. This computation yields a basis $\alpha_j$ of $E$ such that $\alpha_j^\top \tilde{K}_\mu \alpha_i = 0$ for $i < j \le \dim(E)$, and with corresponding positive eigenvalues in decreasing order $\lambda_1, \ldots, \lambda_{\dim(E)}$. Through $v_j = \sum_{i=1}^d \alpha_{j,i} \tilde{\xi}_i$ and writing $r = \dim(E)$, this also yields an orthogonal basis $(v_j)_{i \le r}$ of $\text{span}\{(\tilde{\xi}_i)_{i \le n}\}$, which can be completed to span $\Xi$ through a Gram-Schmidt orthonormalization process using the original basis

$(e_i)_{i\in\mathbb{N}}$. The orthonormal base corresponding to $S_\mu$ is thus $(v_i)_{i\in\mathbb{N}}$, where the $r$ first vectors are the original eigenvectors obtained through the previous maximization. Such a diagonal representation of $S_\mu$ takes the form $S_\mu = S_{\lambda,v}$ where $\lambda = (\lambda_1, ..., \lambda_r, 0, ...)$. This bilinear form is however degenerated on $\Xi$ and facing the same problem encountered in [4, 6] we also propose to solve this issue through a regularization by adding a component $\eta$ on every vector of the base, i.e. defining $\lambda_\eta = (\lambda_1 + \eta, ..., \lambda_r + \eta, \eta, ...)$ with $\eta > 0$, to propose a regularization of $S_\mu$ as:

$$S_{\lambda_\eta,v} = \sum_{i=1}^{r}(\lambda_i + \eta)v_iv_i^* + \sum_{i>r} \eta\, v_iv_i^*.$$

The entropy of a covariance operator $S_{t,e}$ not being defined, we bypass this issue by considering the entropy of its marginal distribution on its first $d$ eigenfunctions, namely introducing the quantity $|S_{t,e}|_d = \frac{d}{2}\ln(2\pi e) + \frac{1}{2}\sum_{i=1}^{d}\ln t_i$. Let us sum up ideas now and consider three normalized measures $\mu, \mu'$ and $\mu'' = \frac{\mu+\mu'}{2}$, which yield three different orthonormal bases $v_i, v_i'$ and $v_i''$ of $\Xi$ and three different families of weights $\lambda_\eta = (\lambda_{i\leq r}+\eta, \eta, ...)$, $\lambda_\eta' = (\lambda_{i\leq r'}' + \eta, \eta, ...)$ and $\lambda_\eta'' = (\lambda_{i\leq r''}'' + \eta, \eta, ...)$. Though working on different bases, those respective $d$ first directions allow us to express an approached form of kernel (2) limited to different subspaces of $\Xi$ of arbitrary size $d \gg r'' \geq \max(r, r')$:

$$
\begin{aligned}
k_{d,\beta}(\mu, \mu') &= \exp\left(-2\beta\left(|S_{\lambda_\eta'',v''}|_d - \frac{|S_{\lambda_\eta,v}|_d + |S_{\lambda_\eta',v'}|_d}{2}\right)\right) \\
&= \left(\frac{\sqrt{\prod_{i=1}^{r} 1 + \frac{\lambda_i}{\eta} \prod_{i=1}^{r'} 1 + \frac{\lambda_i'}{\eta}}}{\prod_{i=1}^{r''} 1 + \frac{\lambda_i''}{\eta}}\right)^{2\beta},
\end{aligned}
\tag{3}
$$

The latter expression is independent of $d$, while letting $d$ go to infinity lets every base on which are computed our entropies span the entire space $\Xi$. Though the latter hint does not establish a valid theoretical proof of the positive definiteness of this kernel, we use this final formula for the following classification experiments.

## 5  Experiments

Following the previous work of [4], we have conducted experiments on an extraction of 500 images ($28 \times 28$ pixels) taken in the MNIST database of handwritten digits, with 50 images for each digit. To each image $z$ we randomly associate a set $\tau(z)$ of 25 to 30 pixels among black points (intensity superior to 191 on a 0 to 255 scale ) in the image, where $\mathcal{X}$ is $\{1, .., 28\} \times \{1, .., 28\}$ in this case. In all our experiments we set $\beta$ to be $\frac{1}{2}$ which always yielded positive definite Gram matrices in practice. To define our RKHS $\Xi$ we used both the linear kernel, $\kappa_a((x_1, y_1), (x_2, y_2)) = (x_1x_2 + y_1y_2)/27^2$ and the Gaussian kernel of width $\sigma$, namely $\kappa_b((x_1, y_1), (x_2, y_2)) = e^{-\frac{(x_1-x_2)^2+(y_1-y_2)^2}{27^2 \cdot 2\sigma^2}}$. The linear case boils down to the simple application presented in the end of section 3 where we fit Gaussian bivariate-laws on our three measures and define similarity through variance analysis. The resulting diagonal variances $(\Sigma_{1,1}, \Sigma_{2,2}), (\Sigma_{1,1}', \Sigma_{2,2}')$ and $(\Sigma_{1,1}'', \Sigma_{2,2}'')$ measure the dispersion of our data for each of the three measures, yielding a kernel value of $\frac{\sqrt{\Sigma_{1,1}\Sigma_{2,2}\Sigma_{1,1}'\Sigma_{2,2}'}}{\Sigma_{1,1}''\Sigma_{2,2}''}$ equal to $0.382$ in the case shown in figure 1. The linear kernel manages a good discrimination between clearly defined digits such as 1 and 0 but fails at doing so when considering numbers whose pixels' distribution cannot be properly characterized by ellipsoid-like shapes. Using instead the Gaussian kernel brings forward a non-linear perspective to the previous approach since it maps now all pixels into Gaussian bells, providing thus a much richer function class for $\Xi$. In this case two parameters

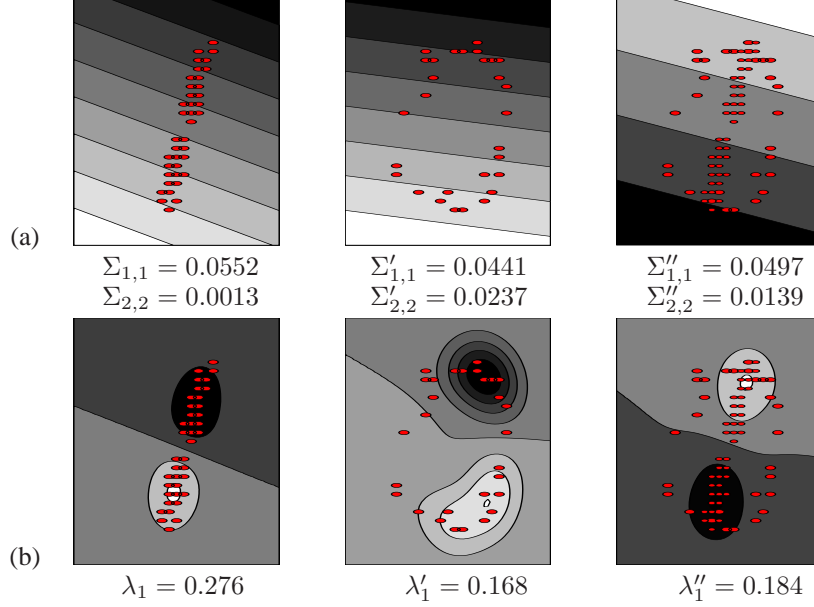

(a)

$$\Sigma_{1,1} = 0.0552 \qquad \Sigma'_{1,1} = 0.0441 \qquad \Sigma''_{1,1} = 0.0497$$
$$\Sigma_{2,2} = 0.0013 \qquad \Sigma'_{2,2} = 0.0237 \qquad \Sigma''_{2,2} = 0.0139$$

(b)

$$\lambda_1 = 0.276 \qquad \lambda'_1 = 0.168 \qquad \lambda''_1 = 0.184$$

Figure 1: First Eigenfunction of three empirical measures $\mu_1$, $\mu_0$ and $\frac{\mu_1+\mu_0}{2}$ using the linear (a) and the Gaussian (b, with $\eta = 0.01, \sigma = 0.1$) kernel. Below each image are the corresponding eigenvalues which correspond to the variance captured by each eigenfunction, the second eigenvalue being also displayed in the linear case (a).

require explicit tuning: $\sigma$ (the width of $\kappa$) controls the range of the typical eigenvalues found in the spectrum of our regularized operators whereas $\eta$ acts as a scaling parameter for the latter values as can be seen in equation (3). An efficient choice can thus only be defined on pairs of parameter, which made us use two ranges of values for $\eta$ and $\sigma$ based on preliminary attempts: $\eta \in 10^{-2} \times \{0.1, 0.3, 0.5, 0.8, 1, 1.5, 2, 3, 5, 8, 10, 20\}$ and $\sigma \in 10^{-1} \times \{0.5, 1, 1.2, 1.5, 1.8, 2, 2.5, 3\}$. For each kernel computed on the base of a $(\sigma, \eta)$ couple, we used a balanced training fold of our dataset to train 10 binary SVM classifiers, namely one for each digit versus all other 9 digits. The class of the remaining images of the test fold was then predicted to be the one with highest SVM score among the the 10 previously trained binary SVMs. Splitting our data into test and training sets was led through a 3-fold cross validation (roughly 332 training images and 168 for testing), averaging the test error on 5 random fold splits of the original data. Those results were obtained using the spider toolbox[1] and graphically displayed in figure (2). Note that the best testing errors were reached using a $\sigma$ value of 0.12 with an $\eta$ parameter within 0.008 and 0.02, this error being roughly 19.5% with a standard deviation inferior to 1% in all the region corresponding to an error lower than 22%. To illustrate the sensibility of our method to the number of sampled points in $\tau$ we show in the same figure the decrease of this error when the number of sampled points ranges from 10 to 30 with independently chosen random points for each computation. As in [4], we also compared our results to the standard RBF kernel on images seen as vectors of $\{0, 1\}^{27 \cdot 27}$, using a fixed number of 30 sampled points and the formula $k(z, z') = e^{-\frac{||z - z'||}{30 \cdot 2\sigma^2}}$. We obtained similar results with an optimal error rate of roughly 44.5% for $\sigma \in \{0.12, 0.15, 0.18\}$. Our results didn't improve by choosing different soft margin $C$ parameters, which we hence just set to be $C = \infty$ as is chosen by default by the spider toolbox.

[1] see http://www.kyb.tuebingen.mpg.de/bs/people/spider/

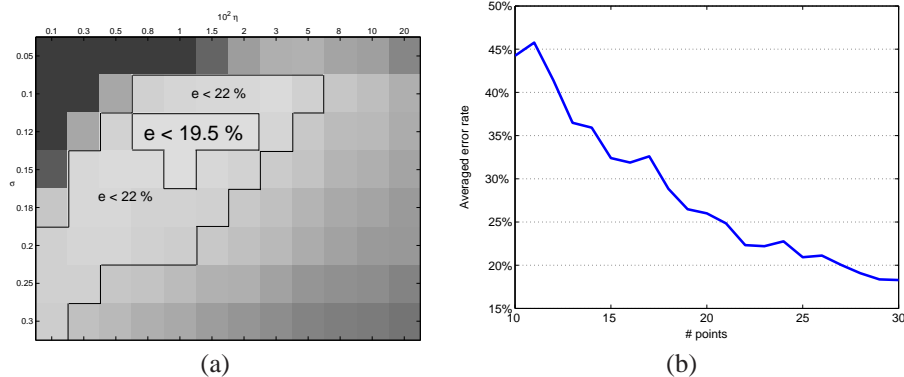

Figure 2: (a) Average test error (displayed as a grey level) of different SVM handwritten character recognition experiments using 500 images from the MNIST database (each seen as a set of 25 to 30 randomly selected black pixels), carried out with 3-fold (2 for training, 1 for test) cross validations with 5 repeats, where parameters $\eta$ (regularization) and $\sigma$ (width of the Gaussian kernel) have been tuned to different values. (b) Curve of the same error (with $\eta = 0.01, \sigma = 0.12$ fixed) depending now on the size of the sets of randomly selected black pixels for each image, this size varying between 10 and 30.

## Acknowledgments

The authors would like to thank Francis Bach, Kenji Fukumizu and Jérémie Jakubowicz for fruitful discussions and Xavier Dupré for his help on the MNIST database.

## References

[1] B. Schölkopf and A.J. Smola. *Learning with Kernels: Support Vector Machines, Regularization, Optimization, and Beyond*. MIT Press, Cambridge, MA, 2002.

[2] J. Lafferty and G. Lebanon. Information diffusion kernels. In *Advances in Neural Information Processing Systems 14*, Cambridge, MA, 2002. MIT Press.

[3] M. Seeger. Covariance kernels from bayesian generative models. In *Advances in Neural Information Processing Systems 14*, pages 905–912, Cambridge, MA, 2002. MIT Press.

[4] R. Kondor and T. Jebara. A kernel between sets of vectors. In *Machine Learning, Proceedings of the Twentieth International Conference (ICML 2003)*, pages 361–368. AAAI Press, 2003.

[5] L. Wolf and A. Shashua. Learning over sets using kernel principal angles. *Journal of Machine Learning Research*, 4:913–931, 2003.

[6] F. Bach and M. Jordan. Kernel independent component analysis. *Journal of Machine Learning Research*, 3:1–48, 2002.

[7] M. Cuturi and J.-P. Vert. A mutual information kernel for sequences. In *IEEE International Joint Conference on Neural Networks*, 2004.

[8] C. Berg, J.P.R. Christensen, and P. Ressel. *Harmonic Analysis on Semigroups*. Springer, 1984.

[9] S. Amari and H. Nagaoka. *Methods of information geometry*. AMS vol. 191, 2001.

[10] F. M. J. Willems, Y. M. Shtarkov, and Tj. J. Tjalkens. The context-tree weighting method: basic properties. *IEEE Transancations on Information Theory*, pages 653–664, 1995.

[11] J.-P. Vert, H. Saigo, and T. Akutsu. Local alignment kernels for protein sequences. In B. Schoelkopf, K. Tsuda, and J.-P. Vert, editors, *Kernel Methods in Computational Biology*. MIT Press, 2004.

[12] T. Cover and J. Thomas. *Elements of Information Theory*. Wiley & Sons, New-York, 1991.

[13] N. Aronszajn. Theory of reproducing kernels. *Transactions of the American Mathematical Society*, 68:337 – 404, 1950.
